# Modelling Uncertainty in the Game of Go

**David H. Stern**
Department of Physics
Cambridge University
dhs26@cam.ac.uk

**Thore Graepel**
Microsoft Research
Cambridge, U.K.
thoreg@microsoft.com

**David J. C. MacKay**
Department of Physics
Cambridge University
mackay@mrao.cam.ac.uk

## Abstract

Go is an ancient oriental game whose complexity has defeated attempts to automate it. We suggest using probability in a Bayesian sense to model the uncertainty arising from the vast complexity of the game tree. We present a simple conditional Markov random field model for predicting the pointwise territory outcome of a game. The topology of the model reflects the spatial structure of the Go board. We describe a version of the Swendsen-Wang process for sampling from the model during learning and apply loopy belief propagation for rapid inference and prediction. The model is trained on several hundred records of professional games. Our experimental results indicate that the model successfully learns to predict territory despite its simplicity.

## 1   Introduction

The game of Go originated in China over 4000 years ago. Its rules are simple (See www.gobase.org for an introduction). Two players, Black and White, take turns to place *stones* on the intersections of an $N \times N$ grid (usually $N = 19$ but smaller boards are in use as well). All the stones of each player are identical. Players place their stones in order to create *territory* by occupying or surrounding areas of the board. The player with the most territory at the end of the game is the winner. A stone is captured if it has been completely surrounded (in the horizontal and vertical directions) by stones of the opponent's colour. Stones in a contiguous 'chain' have the *common fate property*: they are captured all together or not at all [1].

The game that emerges from these simple rules has a complexity that defeats attempts to apply minimax search. The best Go programs play only at the level of weak amateur Go players and Go is therefore considered to be a serious AI challenge not unlike Chess in the 1960s. There are two main reasons for this state of affairs: firstly, the high branching factor of Go (typically 200 to 300 potential moves per position) prevents the expansion of a game tree to any useful depth. Secondly, it is difficult to produce an evaluation function for Go positions. A Go stone has no intrinsic value; its value is determined by its relationships with other stones. Go players evaluate positions using visual pattern recognition and qualitative intuitions which are difficult to formalise.

Most Go programs rely on a large amount of hand-tailored rules and expert knowl-

edge [2]. Some machine learning techniques have been applied to Go with limited success. Schraudolph, Dayan and Sejnowski [3] trained a multi-layer perceptron to evaluate board positions by temporal difference learning. Enzenberger [4] improved on this by structuring the topologies of his neural networks according to the relationships between stones on the board. Graepel et al. [1] made use of the common fate property of chains to construct an efficient graph-based representation of the board. They trained a Support Vector Machine to use this representation to solve Go problems.

Our starting point is the uncertainty about the future course of the game that arises from the vast complexity of the game tree. We propose to explicitly model this uncertainty using probability in a Bayesian sense. The Japanese have a word, *aji*, much used by Go players. Taken literally it means 'taste'. Taste lingers, and likewise the influence of a Go stone lingers (even if it appears weak or dead) because of the uncertainty of the effect it may have in the future. We use a probabilistic model that takes the current board position and predicts for every intersection of the board if it will be Black or White territory. Given such a model the score of the game can be predicted and hence an evaluation function produced. The model is a conditional Markov random field [5] which incorporates the spatial structure of the Go board.

## 2 Models for Predicting Territory

Consider the Go board as an undirected Graph $\mathcal{G} = (\mathcal{N}, \mathcal{E})$ with $N = N_x \times N_y$ nodes $n \in \mathcal{N}$ representing vertices on the board and edges $e \in \mathcal{E}$ connecting vertically and horizontally neighbouring points. We can denote a position as the vector $\mathbf{c} \in \{\text{Black}, \text{White}, \text{Empty}\}^N$ for $c_n = c(n)$ and similarly the final territory outcome of the game as $\mathbf{s} \in \{+1, -1\}^N$ for $s_n = s(n)$. For convenience we score from the point of view of Black so elements of $\mathbf{s}$ representing Black territory are valued $+1$ and elements representing white territory are valued $-1$. Go players will note that we are adopting the Chinese method of scoring empty as well as occupied intersections. The distribution we wish to model is $P(\mathbf{s}|\mathbf{c})$, that is, the distribution over final territory outcomes given the current position. Such a model would be useful for several reasons.

- Most importantly, the detailed outcomes provide us with a simple evaluation function for Go positions by the expected score, $u(\mathbf{c}) := \langle \sum_i s_i \rangle_{P(\mathbf{s}|\mathbf{c})}$. An alternative (and probably better) evaluation function is given by the probability of winning which takes the form $P(\text{Black wins}) = P(\sum_i s_i > \text{komi})$, where *komi* refers to the winning threshold for Black.

- *Connectivity* of stones is vital because stones can draw strength from other stones. Connectivity could be measured by the correlation between nodes under the distribution $P(\mathbf{s}|\mathbf{c})$. This would allow us to segment the board into 'groups' of stones to reduce complexity.

- It would also be useful to observe cases where we have an anti-correlation between nodes in the territory prediction. Japanese refer to such cases as *miai* in which only one of two desired results can be achieved at the expense of the other - a consequence of moving in turns.

- The fate of a group of Go stones could be estimated from the distribution $P(\mathbf{s}|\mathbf{c})$ by marginalising out the nodes not involved.

The way stones exert long range influence can be considered recursive. A stone influences its neighbours, who influence their neighbours and so on. A simple model

which exploits this idea is to consider the Go board itself as an undirected graphical model in the form of a Conditional Random Field (CRF) [5]. We factorize the distribution as

$$P(\mathbf{s}|\mathbf{c}) = \frac{1}{Z(\mathbf{c}, \boldsymbol{\theta})} \prod_{f \in \mathcal{F}} \psi_f(\mathbf{s}_f, \mathbf{c}_f, \boldsymbol{\theta}_f) = \frac{1}{Z(\mathbf{c}, \boldsymbol{\theta})} \exp\left(\sum_{f \in \mathcal{F}} \log(\psi_f(\mathbf{s}_f, \mathbf{c}_f, \boldsymbol{\theta}_f))\right).$$
(1)

The simplest form of this model has one factor for each pair of neighbouring nodes $i, j$ so $\psi_f(\mathbf{s}_f, \mathbf{c}_f, \boldsymbol{\theta}_f) = \psi_f(s_i, s_j, c_i, c_j, \boldsymbol{\theta}_f)$.

**Boltzmann5**  For our first model we decompose the factors into 'coupling' terms and 'external field' terms as follows:

$$P(\mathbf{s}|\mathbf{c}) = \frac{1}{Z(\mathbf{c}, \boldsymbol{\theta})} \exp\left(\sum_{(i,j) \in \mathcal{F}} \{w(c_i, c_j)s_i s_j + h(c_i)s_i + h(c_j)s_j\}\right)$$
(2)

This gives a Boltzmann machine whose connections have the grid topology of the board. The couplings between territory-outcome nodes depend on the current board position local to those nodes and the external field at each node is determined by the state of the board at that location. We assume that Go positions with their associated territory positions are symmetric with respect to colour reversal so $\psi_f(s_i, s_j, c_i, c_j, \boldsymbol{\theta}_f) = \psi_f(-s_i, -s_j, -c_i, -c_j, \boldsymbol{\theta}_f)$. Pairwise connections are also invariant to direction reversal so $\psi_f(s_i, s_j, c_i, c_j, \boldsymbol{\theta}_f) = \psi_f(s_j, s_i, c_j, c_i, \boldsymbol{\theta}_f)$. It follows that the model described in 2 can be specified by just five parameters:

- $w_{\text{chains}} = w(\text{Black}, \text{Black}) = w(\text{White}, \text{White})$,
- $w_{\text{inter-chain}} = w(\text{Black}, \text{White}) = w(\text{White}, \text{Black})$,
- $w_{\text{chain-empty}} = w(\text{Empty}, \text{White}) = w(\text{Empty}, \text{Black})$,
- $w_{\text{empty}} = w(\text{Empty}, \text{Empty})$,
- $h_{\text{stones}} = h(\text{Black}) = -h(\text{White})$,

and $h(\text{empty})$ is set to zero by symmetry. We will refer to this model as *Boltzmann5*. This simple model is interesting because all these parameters are readily interpreted. For example we would expect $w_{\text{chains}}$ to take on a large positive value since chains have common fate.

**BoltzmannLiberties**  A feature that has particular utility for evaluating Go positions is the number of *liberties* associated with a chain of stones. A liberty of a chain is an empty vertex adjacent to it. The number of liberties indicates a chain's safety because the opponent would have to occupy all the liberties to capture the chain. Our second model takes this information into account:

$$P(\mathbf{s}|\mathbf{c}) = \frac{1}{Z(\mathbf{c}, \boldsymbol{\theta})} \exp\left(\sum_{(i,j) \in \mathcal{F}} w(c_i, c_j, s_i, s_j, l_i, l_j)\right),$$
(3)

where $l_i$ is element $i$ of a vector $\mathbf{l} \in \{+1, +2, +3, 4 \text{ or more}\}^N$ the liberty count of each vertex on the Go board. A group with four or more liberties is considered relatively safe. Again we can apply symmetry arguments and end up with 78 parameters. We will refer to this model as *BoltzmannLiberties*.

We trained the two models using board positions from a database of 22,000 games between expert Go players[1]. The territory outcomes of a subset of these games

[1]The GoGoD database, April 2003. URL:http://www.gogod.demon.co.uk

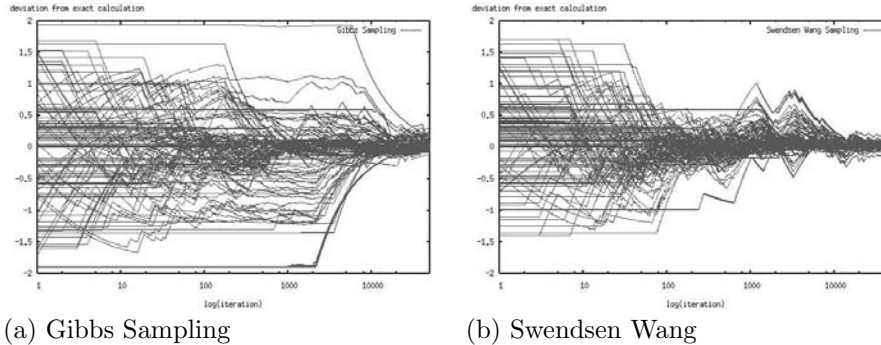

(a) Gibbs Sampling          (b) Swendsen Wang

Figure 1: Comparing ordinary Gibbs with Swendsen Wang sampling for *Boltzmann5*. Shown are the differences between the running averages and the exact marginals for each of the 361 nodes plotted as a function of the number of whole-board samples.

were determined using the Go program GnuGo[2] to analyse their final positions. Each training example comprised a board position $\mathbf{c}$, with its associated territory outcome $\mathbf{s}$. Training was performed by maximising the likelihood $\ln P(\mathbf{s}'|\mathbf{c})$ using gradient descent. In order to calculate the likelihood it is necessary to perform inference to obtain the marginal expectations of the potentials.

## 3   Inference Methods

It is possible to perform exact inference on the model by variable elimination [6]. Eliminating nodes one diagonal at a time gave an efficient computation. The cost of exact inference was still too high for general use but it was used to compare other inference methods.

**Sampling**   The standard method for sampling from a Boltzmann machine is to use Gibbs sampling where each node is updated one at a time, conditional on the others. However, Gibbs sampling mixes slowly for spin systems with strong correlations. A generalisation of the Swendsen-Wang process [7] alleviates this problem. The original Swendsen-Wang algorithm samples from a ferromagnetic Ising model with no external field by adding an additional set of 'bond' nodes $\mathbf{d}$, one attached to each factor (edge) in the original graph. Each of these nodes can either be in the state 'bond' or 'no bond'. The new factor potentials $\psi_f(\mathbf{s}_f, \mathbf{c}_f, \mathbf{d}_f, \boldsymbol{\theta}_f)$ are chosen such that if a bond exists between a pair of spins then they are forced to be in the same state. Conditional on the bonds, each cluster has an equal probability of having all its spins in the 'up' state or all in the 'down' state. The algorithm samples from $P(\mathbf{s}|\mathbf{d}, \mathbf{c}, \boldsymbol{\theta})$ and $P(\mathbf{d}|\mathbf{s}, \mathbf{c}, \boldsymbol{\theta})$ in turn (flipping clusters and forming bonds respectively). It can be generalised to models with arbitrary couplings and biases [7, 8]. The new factor potentials $\psi_f(\mathbf{s}_f, \mathbf{c}_f, \mathbf{d}_f, \boldsymbol{\theta}_f)$ have the following effect: if the coupling is positive then when the $\mathbf{d}$ node is in the 'bond' state it forces the two spins to be in the same state; if the coupling is negative the 'bond' state forces the two spins to be opposite. The probability of each cluster being in each state depends on the sum of the biases involved. Figure 1 shows that the mixing rate of the sampling process is improved by using Swendsen-Wang allowing us to find accurate marginals for a single position in a couple of seconds.

**Loopy Belief Propagation** In order to perform very rapid (approximate) inference we used the loopy belief propagation (BP) algorithm [9] and the results are examined in Section 4. This algorithm is similar to an *influence function* [10], as often used by Go programmers to segment the board into Black and White territory and for this reason is laid out below.

For each board vertex $j \in \mathcal{N}$, create a data structure called a *node* containing:

1. $\mathcal{A}(j)$, the set of nodes corresponding to the neighbours of vertex $j$,
2. a set of *new messages* $m_{ij}^{new}(s_j) \in \mathcal{M}_{new}$, one for each $i \in \mathcal{A}(j)$,
3. a set of *old messages* $m_{ij}^{old}(s_j) \in \mathcal{M}_{old}$, one for each $i \in \mathcal{A}(j)$,
4. a *belief* $b_j(s_j)$.

---

**repeat**
  **for all** $j \in \mathcal{N}$ **do**
    **for all** $i \in \mathcal{A}(j)$ **do**
      **for all** $s_j \in \{\text{Black}, \text{White}\}$ **do**
        let variable SUM := 0,
        **for all** $s_i \in \{\text{Black}, \text{White}\}$ **do**
          $\text{SUM} := \text{SUM} + \psi_{(i,j)}(s_i, s_j) \prod_{q \in \mathcal{A}(i) \backslash j} m_{qi}^{old}(s_i),$
        **end for**
        $m_{ij}^{new}(s_j) := \text{SUM},$
      **end for**
    **end for**
  **end for**
  **for all** messages, $m_{xy}^{new}(s_y) \in \mathcal{M}_{new}$ **do**
    $m_{xy}^{new}(s_y) := \lambda m_{xy}^{old}(s_y) + (1 - \lambda) m_{xy}^{new}(s_y),$
  **end for**
**until** completed I iterations (typically I=50)

Belief Update:
**for all** $j \in \mathcal{N}$ **do**
  **for all** $s_j \in \{\text{Black}, \text{White}\}$ **do**
    $b_j(s_j) := \prod_{q \in \mathcal{A}(j)} m_{qj}^{new}(s_j)$
  **end for**
**end for**

---

Here, $\lambda$ (typically 0.5), damps any oscillations. $\psi_{(i,j)}(s_i, s_j)$ is the factor potential (see (1)) and in the case of *Boltzmann5* takes on the form $\psi_{(i,j)}(s_i, s_j) = \exp(w(c_i, c_j)s_i s_j + h(c_i)s_i + h(c_j)s_j)$. Now the probability of each vertex being Black or White territory is found by normalising the beliefs at each node. For example $P(s_j = \text{Black}) = b_j(\text{Black})/Z$ where $Z = b_j(\text{Black}) + b_j(\text{White})$. The accuracy of the loopy BP approximation appears to be improved by using it during the parameter learning stage in cases where it is to be used in evaluation.

## 4 Results for Territory Prediction

**Some Learnt Parameters** Here are some parameters learnt for the *Boltzmann5* model (2). This model was trained on 290 positions from expert Go games at move 80. Training was performed by maximum likelihood as described in Section 2.

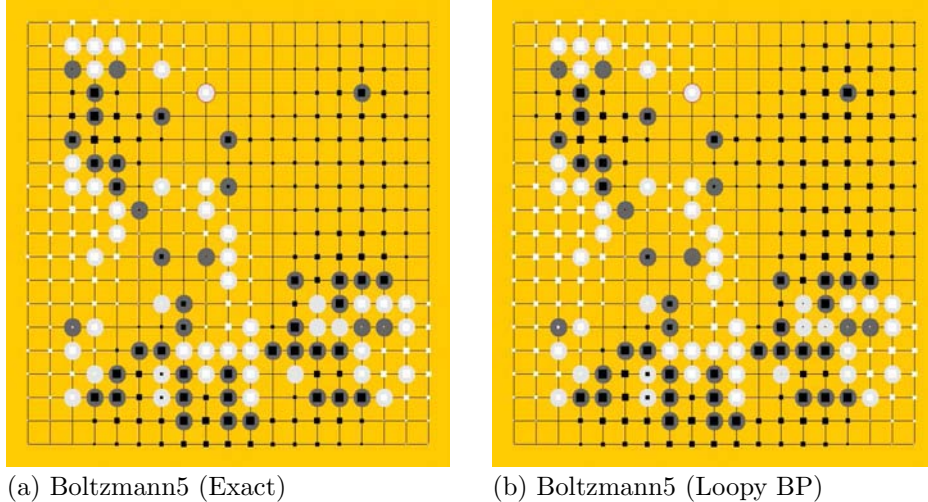

(a) Boltzmann5 (Exact)        (b) Boltzmann5 (Loopy BP)

Figure 2: Comparing territory predictions for a Go position from a professional game at move 90. The circles represent stones. The small black and white squares at each vertex represent the average territory prediction at that vertex, from $-1$ (maximum white square) to $+1$ (maximum black square).

- $h_{\text{stones}} = 0.265$
- $w_{\text{empty}} = 0.427$
- $w_{\text{chain}-\text{empty}} = 0.442$
- $w_{\text{chains}} = 2.74$
- $w_{\text{inter}-\text{chain}} = 0.521$

The values of these parameters can be interpreted. For example $w_{\text{chains}}$ corresponds to the correlation between the likely territory outcome of two adjacent vertices in a chain of connected stones. The high value of this parameter derives from the 'common fate' property of chains as described in Section 1.

Interestingly, the value of the parameter $w_{\text{empty}}$ (corresponding to the coupling between territory predictions of neighbouring vertices in empty space) is 0.427 which is close to the critical coupling for an Ising model, 0.441.

**Territory Predictions** Figure 2 gives examples of territory predictions generated by *Boltzmann5*. In comparison, Figure 3 shows the prediction of *BoltzmannLiberties* and a territory prediction from *The Many Faces of Go* [2]. Go players confirm that the territory predictions produced by the models are reasonable, even around loose groups of Black and White stones. Compare Figures 2 (a) and 3 (a); when liberty counts are included as features, the model can more confidently identify which of the two small chains competing in the bottom right of the board is dead. Comparing Figure 2 (a) and (b) Loopy BP appears to give over-confident predictions in the top right of the board where few stones are present. However, it is a good approximation where many stones are present (bottom left).

**Comparing Models and Inference Methods** Figure 4 shows cross-entropies between model territory predictions and true final territory outcomes for a dataset of expert games. As we progress through a game, predictions become more accurate (not surprising) but the spread of the accuracy increases, possibly due to incorrect assessment of the life-and-death status of groups. Swendsen-Wang performs better than Loopy BP, which may suffer from its over-confidence. *BoltzmannLiberties* performs better than *Boltzmann5* (when using Swendsen-Wang) the difference in

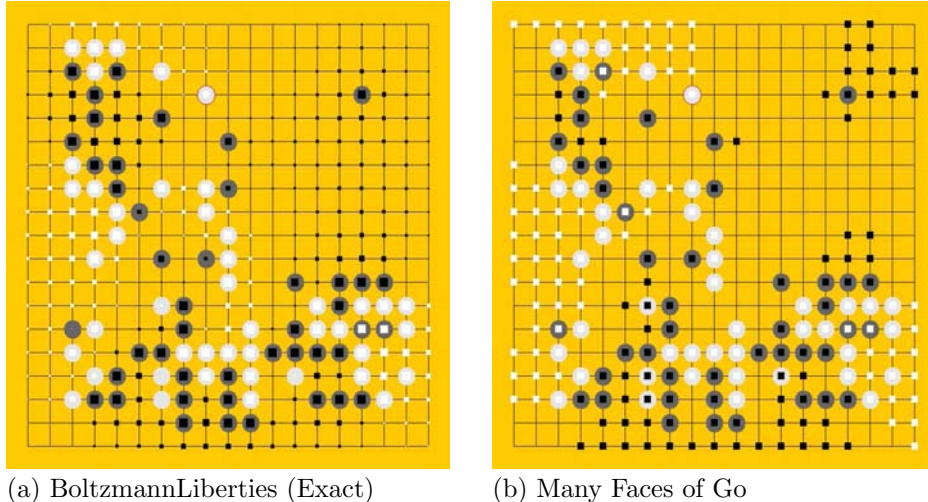

(a) BoltzmannLiberties (Exact)          (b) Many Faces of Go

Figure 3: Diagram (a) is produced by exact inference (training was also by Loopy BP). Diagram (b) shows the territory predicted by The Many Faces of Go (MFG) [2]. MFG uses of a rule-based expert system and its prediction for each vertex has three possible values: 'White', 'Black' or 'unknown/neutral'.

performance increasing later in the game when liberty counts become more useful.

## 5  Modelling Move Selection

In order to produce a Go playing program we are interested in modelling the selection of moves. A measure of performance of such a model is the likelihood it assigns to professional moves as measured by

$$\sum_{\text{games}} \sum_{\text{moves}} \log P(\text{move}|\text{model}). \tag{4}$$

We can obtain a probability over moves by choosing a Gibbs distribution with the negative energy replaced by the evaluation function,

$$P(\text{move}|\text{model}, \mathbf{w}) = \frac{e^{\beta u(\mathbf{c}', \mathbf{w})}}{Z(\mathbf{w})} \tag{5}$$

where $u(\mathbf{c}', \mathbf{w})$ is an evaluation function evaluated at the board position $\mathbf{c}'$ resulting from a given move. The inverse temperature parameter $\beta$ determines the degree to which the move made depends on its evaluation. The territory predictions from the models *Boltzmann5* and *BoltzmannLiberties* can be combined with the evaluation function of Section 2 to produce position evaluators.

## 6  Conclusions

We have presented a probabilistic framework for modelling uncertainty in the game of Go. A simple model which incorporates the spatial structure of a board position can perform well at predicting the territory outcomes of Go games. The models described here could be improved by extracting more features from board positions and increasing the size of the factors (see (1)).

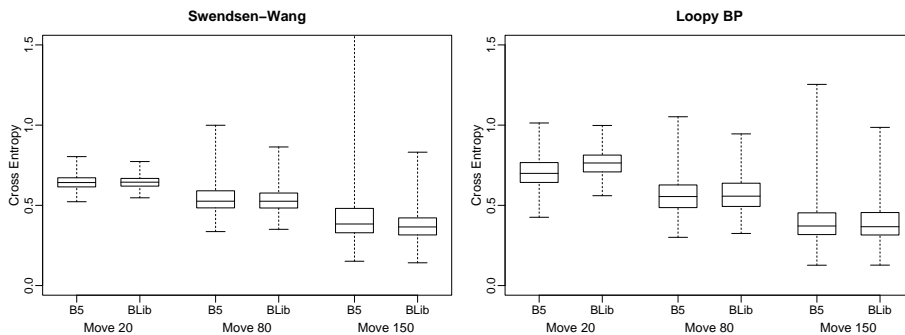

Figure 4: Cross entropies $\frac{1}{N}\sum_n^N [s'_n \log s_n + (1-s'_n)\log(1-s_n)]$ between actual and predicted territory outcomes, $s'_n$ and $n$ for 327 Go positions. Sampling is compared with Loopy BP (training and testing). 3 board positions were analysed for each game (moves 20, 80 and 150). The Boltzmann5 (B5) and the BoltzmannLiberties (BLib) models are compared.

**Acknowledgements**   We thank I. Murray for helpful discussions on sampling and T. Minka for general advice about probabilistic inference. This work was supported by a grant from Microsoft Research UK.

## Footnotes

[2]URL:http://www.gnu.org/software/gnugo/gnugo.html

# References

[1] Thore Graepel, Mike Goutrie, Marco Kruger, and Ralf Herbrich. Learning on graphs in the game of Go. In *Proceedings of the International Conference on Artificial Neural Networks, ICANN 2001*, 2001.

[2] David Fotland. Knowledge representation in the many faces of go. URL: ftp://www.joy.ne.jp/welcome/igs/Go/computer/mfg.tex.Z, 1993.

[3] Nicol N. Schrauldolph, Peter Dayan, and Terrance J. Sejnowski. Temporal difference learning of position evaluation in the game of go. In *Advances in Neural Information Processing Systems 6*, pages 817–824, San Fransisco, 1994. Morgan Kaufmann.

[4] Markus Enzenberger. The integration of a priori knowledge into a Go playing neural network. URL: http://www.markus-enzenberger.de/neurogo.html, 1996.

[5] John Lafferty, Andrew McCallum, and Fernando Pereira. Conditional random fields: Probabilistic models for segmenting and labeling sequence data. In *Proc. Int. Conf. on Machine Learning*, 2001.

[6] Fabio Gagliardi Cozman. Generalizing variable elimination in Bayesian networks. In *Proceedings of the IBERAMIA/SBIA 2000 Workshops*, pages 27–32, 2000.

[7] R. H. Swendsen and J-S Wang. Nonuniversal critical dynamics in Monte Carlo simulations. *Physical Review Letters*, 58:86–88, 1987.

[8] Robert G. Edwards and Alan D. Sokal. Generalisation of the Fortuin-Kasteleyn-Swendsen-Wang representation and Monte Carlo algorithm. *Physical Review Letters*, 38(6), 1988.

[9] Yair Weiss. Belief propagation and revision in networks with loops. Technical report, AI Lab Memo, MIT, Cambridge, 1998.

[10] A. L. Zobrist. *Feature Extractions and Representations for Pattern Recognition and the Game of Go*. PhD thesis, Graduate School of the University of Wisconsin, 1970.